# Induction of Multiscale Temporal Structure

**Michael C. Moser**
Department of Computer Science &
Institute of Cognitive Science
University of Colorado
Boulder, CO 80309–0430

## Abstract

Learning structure in temporally-extended sequences is a difficult computational problem because only a fraction of the relevant information is available at any instant. Although variants of back propagation can in principle be used to find structure in sequences, in practice they are not sufficiently powerful to discover arbitrary contingencies, especially those spanning long temporal intervals or involving high order statistics. For example, in designing a connectionist network for music composition, we have encountered the problem that the net is able to learn musical structure that occurs locally in time—e.g., relations among notes within a musical phrase—but not structure that occurs over longer time periods—e.g., relations among phrases. To address this problem, we require a means of constructing a *reduced description* of the sequence that makes global aspects more explicit or more readily detectable. I propose to achieve this using hidden units that operate with different time constants. Simulation experiments indicate that slower time-scale hidden units are able to pick up global structure, structure that simply can not be learned by standard back propagation.

Many patterns in the world are intrinsically temporal, e.g., speech, music, the unfolding of events. Recurrent neural net architectures have been devised to accommodate time-varying sequences. For example, the architecture shown in Figure 1 can map a sequence of inputs to a sequence of outputs. Learning structure in temporally-extended sequences is a difficult computational problem because the input pattern may not contain all the task-relevant information at any instant. Thus,

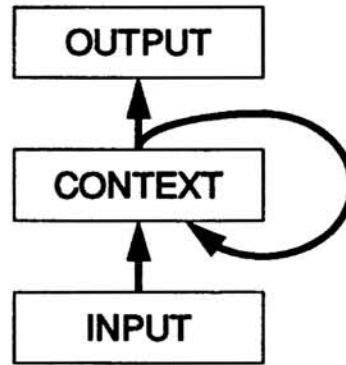

Figure 1: A generic recurrent network architecture for processing input and output sequences. Each box corresponds to a layer of units, each line to full connectivity between layers.

the context layer must hold on to relevant aspects of the input history until a later point in time at which they can be used.

In principle, variants of back propagation for recurrent networks (Rumelhart, Hinton, & Williams, 1986; Williams & Zipser, 1989) can discover an appropriate representation in the context layer for a particular task. In practice, however, back propagation is not sufficiently powerful to discover arbitrary contingencies, especially those that span long temporal intervals or that involve high order statistics (e.g., Mozer, 1989; Rohwer, 1990; Schmidhuber, 1991).

Let me present a simple situation where back propagation fails. It involves remembering an event over an interval of time. A variant of this task was first studied by Schmidhuber (1991). The input is a sequence of discrete symbols: A, B, C, D, $\cdots$, X, Y. The task is to predict the next symbol in the sequence. Each sequence begins with either an X or a Y—call this the *trigger symbol*—and is followed by a fixed sequence such as ABCDE, which in turn is followed by a second instance of the trigger symbol, i.e., XABCDEX or or YABCDEY. To perform the prediction task, it is necessary to store the trigger symbol when it is first presented, and then to recall the same symbol five time steps later.

The number of symbols intervening between the two triggers—call this the *gap*—can be varied. By training different networks on different gaps, we can examine how difficult the learning task is as a function of gap. To better control the experiments, all input sequences had the same length and consisted of either X or Y followed by ABCDEFGHIJK. The second instance of the trigger symbol was inserted at various points in the sequence. For example, XABCDXEFGHIJK represents a gap of 4, YABCDEFGHYIJK a gap of 8.

Each training set consisted of two sequences, one with X and one with Y. Different networks were trained on different gaps. The network architecture consisted of one input and output unit per symbol, and ten context units. Twenty-five replications of each network were run with different random initial weights. If the training set was not learned within 10000 epochs, the replication was counted as a "failure." The primary result was that training sets with gaps of 4 or more could not be learned reliably, as shown in Table 1.

Table 1: Learning contingencies across gaps

| gap | % failures | mean # epochs to learn |
|-----|-----------|------------------------|
| 2   | 0         | 468                    |
| 4   | 36        | 7406                   |
| 6   | 92        | 9830                   |
| 8   | 100       | 10000                  |
| 10  | 100       | 10000                  |

The results are suprisingly poor. My general impression is that back propagation is powerful enough to learn only structure that is fairly *local* in time. For instance, in earlier work on neural net music composition (Mozer & Soukup, 1991), we found that our network could master the rules of composition for notes within a musical phrase, but not rules operating at a more global level—rules for how phrases are interrelated.

The focus of the present work is on devising learning algorithms and architectures for better handling temporal structure at more *global* scales, as well as multiscale or hierarchical structure. This difficult problem has been identified and studied by several other researchers, including Miyata and Burr (1990), Rohwer (1990), and Schmidhuber (1991).

# 1   BUILDING A REDUCED DESCRIPTION

The basic idea behind my work involves building a *reduced description* (Hinton, 1988) of the sequence that makes global aspects more explicit or more readily detectable. The challenge of this approach is to devise an appropriate reduced description. I've experimented with a scheme that constructs a reduced description that is essentially a bird's eye view of the sequence, sacrificing a representation of individual elements for the overall contour of the sequence. Imagine a musical tape played at double the regular speed. Individual sounds are blended together and become indistinguishable. However, coarser time-scale events become more explicit, such as an ascending trend in pitch or a repeated progression of notes. Figure 2 illustrates the idea. The curve in the left graph, depicting a sequence of individual pitches, has been smoothed and compressed to produce the right graph. Mathematically, "smoothed and compressed" means that the waveform has been low-pass filtered and sampled at a lower rate. The result is a waveform in which the alternating upwards and downwards flow is unmistakable.

Multiple views of the sequence are realized using context units that operate with different *time constants*:

$$c_i(t) = \tau_i c_i(t-1) + (1 - \tau_i) \tanh[net_i(t)], \tag{1}$$

where $c_i(t)$ is the activity of context unit $i$ at time $t$, $net_i(t)$ is the net input to unit $i$ at time $t$, including activity both from the input layer and the recurrent context connections, and $\tau_i$ is a time constant associated with each unit that has the range $(0, 1)$ and determines the responsiveness of the unit—the rate at which

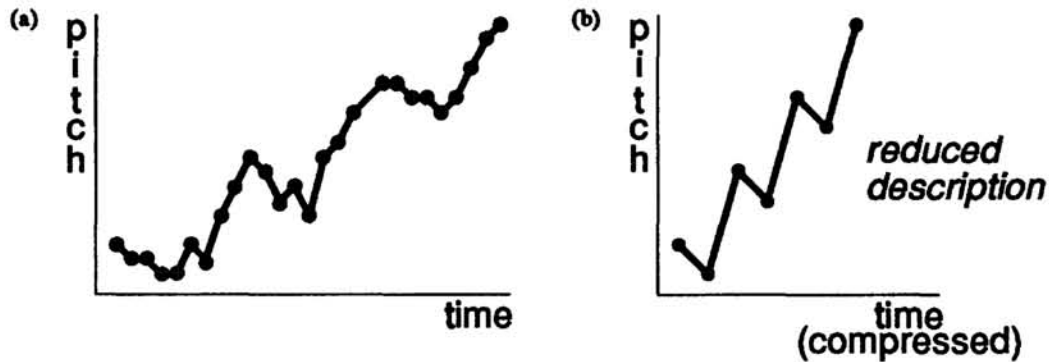

Figure 2: (a) A sequence of musical notes. The vertical axis indicates the pitch, the horizontal axis time. Each point corresponds to a particular note. (b) A smoothed, compact view of the sequence.

its activity changes. With $\tau_i = 0$, the activation rule reduces to the standard one and the unit can sharply change its response based on a new input. With large $\tau_i$, the unit is sluggish, holding on to much of its previous value and thereby averaging the response to the net input over time. At the extreme of $\tau_i = 1$, the second term drops out and the unit's activity becomes fixed. Thus, large $\tau_i$ smooth out the response of a context unit over time. Note, however, that what is smoothed is the activity of the context units, not the input itself as Figure 2 might suggest.

Smoothing is one property that distinguishes the waveform in Figure 2b from the original. The other property, compactness, is also achieved by a large $\tau_i$, although somewhat indirectly. The key benefit of the compact waveform in Figure 2b is that it allows a longer period of time to be viewed in a single glance, thereby explicating contingencies occurring in this interval during learning. The context unit activation rule (Equation 1) permits this. To see why this is the case, consider the relation between the error derivative with respect to the context units at time $t$, $\partial E/\partial c(t)$, and the error back propagated to the previous step, $t - 1$. One contribution to $\partial E/\partial c_i(t - 1)$, from the first term in Equation 1, is

$$\frac{\partial E}{\partial c_i(t)} \frac{\partial}{\partial c_i(t-1)} \left[ \tau_i c_i(t-1) \right] = \tau_i \frac{\partial E}{\partial c_i(t)}. \tag{2}$$

This means that when $\tau_i$ is large, most of the error signal in context unit $i$ at time $t$ is carried back to time $t - 1$. Intuitively, just as the activation of units with large $\tau_i$ changes slowly forward in time, the error propagated back through these units changes slowly too. Thus, the back propagated error signal can make contact with points further back in time, facilitating the learning of more global structure in the input sequence.

Time constants have been incorporated into the activation rules of other connectionist architectures (Jordan, 1987; McClelland, 1979; Mozer, 1989; Pearlmutter, 1989; Pineda, 1987). However, none of this work has exploited time constants to control the temporal responsivity of individual units.

## 2   LEARNING AABA PHRASE PATTERNS

A simple simulation illustrates the benefits of temporal reduced descriptions. I generated pseudo musical phrases consisting of five notes in ascending chromatic order, e.g., F#$_2$ G$_2$ G#$_2$ A$_2$ A#$_2$ or C$_4$ C#$_4$ D$_4$ D#$_4$ E$_4$, where the first pitch was selected at random.[1] Pairs of phrases—call them *A* and *B*—were concatenated to form an *AABA* pattern, terminated by a special END marker. The complete melody then consisted of 21 elements—four phrases of five notes followed by the END marker—an example of which is:

F#$_2$ G$_2$ G#$_2$ A$_2$ A#$_2$ F#$_2$ G$_2$ G#$_2$ A$_2$ A#$_2$ C$_4$ C#$_4$ D$_4$ D#$_4$ E$_4$ F#$_2$ G$_2$ G#$_2$ A$_2$ A#$_2$ END.

Two versions of CONCERT were tested, each with 35 context units. In the *standard* version, all 35 units had $\tau = 0$; in the *reduced description* or *RD* version, 30 had $\tau = 0$ and 5 had $\tau = 0.8$. The training set consisted of 200 examples and the test set another 100 examples. Ten replications of each simulation were run for 300 passes through the training set. See Mozer and Soukup (1991) for details of the network architecture and note representations.

Because of the way that the sequences are organized, certain pitches can be predicted based on local structure whereas other pitches require a more global memory of the sequence. In particular, the second through fifth pitches within a phrase can be predicted based on knowledge of the immediately preceding pitch. To predict the first pitch in the repeated A phrases and to predict the END marker, more global information is necessary. Thus, the analysis was split to distinguish between pitches requiring only local structure and pitches requiring more global structure. As Table 2 shows, performance requiring global structure was significantly better for the RD version ($F(1,9)=179.8$, $p < .001$), but there was only a marginally reliable difference for performance involving local structure ($F(1,9)=3.82$, $p=.08$). The global structure can be further broken down to prediction of the END marker and prediction of the first pitch of the repeated A phrases. In both cases, the performance improvement for the RD version was significant: 88.0% versus 52.9% for the end of sequence ($F(1,9)=220$, $p < .001$); 69.4% versus 61.2% for the first pitch ($F(1,9)=77.6$, $p < .001$).

Experiments with different values of $\tau$ in the range .7–.95 yielded qualitatively similar results, as did experiments in which the A and B phrases were formed by random walks in the key of C major.

Table 2: Performance on AABA phrases

| structure | standard version | RD version |
|---|---|---|
| local | 97.3% | 96.7% |
| global | 58.4% | 75.6% |

# 3    DETECTING CONTINGENCIES ACROSS GAPS— REVISITED

I now return to the prediction task involving sequences containing two X's or Y's separated by a stream of intervening symbols. A reduced description network had no problem learning the contingency across wide gaps. Table 3 compares the results presented earlier for a standard net with ten context units and the results for an RD net having six standard context units ($\tau = 0$) and four units having identical nonzero $\tau$, in the range of .75–.95. More on the choice of $\tau$ below, but first observe that the reduced description net had a 100% success rate. Indeed, it had no difficulty with much wider gaps: I tested gaps of up to 25 symbols. The number of epochs to learn scales roughly linearly with the gap.

When the task was modified slightly such that the intervening symbols were randomly selected from the set {A,B,C,D}, the RD net still had no difficulty with the prediction task.

The bad news here is that the choice of $\tau$ can be important. In the results reported above, $\tau$ was selected to optimize performance. In general, a larger $\tau$ was needed to span larger gaps. For small gaps, performance was insensitive to the particular $\tau$ chosen. However, the larger the temporal gap that had to be spanned, the smaller the range of $\tau$ values that gave acceptable results. This would appear to be a serious limitation of the approach. However, there are several potential solutions.

1. One might try using back propagation to train the time constants directly. This does not work particularly well on the problems I've examined, apparently because the path to an appropriate $\tau$ is fraught with local optima. Using gradient descent to fine tune $\tau$, once it's in the right neighborhood, is somewhat more successful.

2. One might include a complete range of $\tau$ values in the context layer. It is not difficult to determine a rough correspondence between the choice of $\tau$ and the temporal interval to which a unit is optimally tuned. If sufficient units are used to span a range of intervals, the network should perform well. The down side, of course, is that this gives the network an excess of weight parameters with which it could potentially overfit the training data. However, because the different $\tau$ correspond to different temporal scales, there is much less freedom to abuse the weights here than, say, in a situation where additional hidden units are added to a feedforward network.

Table 3: Learning contingencies across gaps (revisited)

| | standard net | | reduced description net | |
|---|---|---|---|---|
| gap | % failures | mean # epochs to learn | % failures | mean # epochs to learn |
| 2 | 0 | 468 | 0 | 328 |
| 4 | 36 | 7406 | 0 | 584 |
| 6 | 92 | 9830 | 0 | 992 |
| 8 | 100 | 10000 | 0 | 1312 |
| 10 | 100 | 10000 | 0 | 1630 |

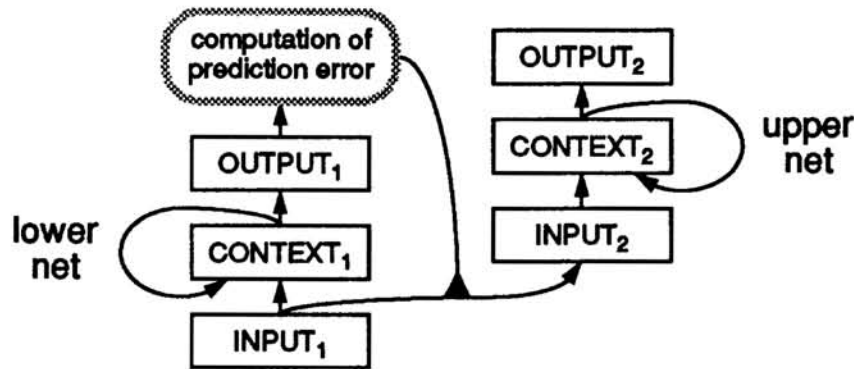

Figure 3: A sketch of the Schmidhuber (1991) architecture

3. One might dynamically adjust $\tau$ as a sequence is presented based on external criteria. In Section 5, I discuss one such criterion.

## 4    MUSIC COMPOSITION

I have used music composition as a domain for testing and evaluating different approaches to learning multiscale temporal structure. In previous work (Mozer & Soukup, 1991), we designed a sequential prediction network, called CONCERT, that learns to reproduce a set of pieces of a particular musical style. CONCERT also learns structural regularities of the musical style, and can be used to compose new pieces in the same style. CONCERT was trained on a set of Bach pieces and a set of traditional European folk melodies. The compositions it produces were reasonably pleasant, but were lacking in global coherence. The compositions tended to wander randomly with little direction, modulating haphazardly from major to minor keys, flip-flopping from the style of a march to that of a minuet. I attribute these problems to the fact that CONCERT had learned only local temporal structure.

I have recently trained CONCERT on a third set of examples—waltzes—and have included context units that operate with a range of time constants. There is a consensus among listeners that the new compositions are more coherent. I am presently running more controlled simulations using the same musical training set and versions of CONCERT with and without reduced descriptions, and am attempting to quantify CONCERT's abilities at various temporal scales.

## 5    A HYBRID APPROACH

Schmidhuber (1991; this volume) has proposed an alternative approach to learning multiscale temporal structure in sequences. His approach, the *chunking architecture*, basically involves two (or more) sequential prediction networks cascaded together (Figure 3). The lower net receives each input and attempts to predict the next input. When it fails to predict reliably, the next input is passed to the upper net. Thus, once the lower net has been trained to predict local temporal structure, such structure is removed from the input to the upper net. This simplifies the task of learning global structure in the upper net.

Schmidhuber's approach has some serious limitations, as does the approach I've described. We have thus merged the two in a scheme that incorporates the strengths of each approach (Schmidhuber, Prelinger, Mozer, Blumenthal, & Mathis, in preparation). The architecture is the same as depicted in Figure 3, except that all units in the upper net have associated with them a time constant $\tau_u$, and the prediction error in the lower net determines $\tau_u$. In effect, this allows the upper net to kick in only when the lower net fails to predict. This avoid the problem of selecting time constants, which my approach suffers. This also avoids the drawback of Schmidhuber's approach that yes-or-no decisions must be made about whether the lower net was successful. Initial simulation experiments indicate robust performance of the hybrid algorithm.

## Acknowledgements

This research was supported by NSF Presidential Young Investigator award IRI–9058450, grant 90–21 from the James S. McDonnell Foundation, and DEC external research grant 1250. Thanks to Jürgen Schmidhuber and Paul Smolensky for helpful comments regarding this work, and to Darren Hardy for technical assistance.

## Footnotes

[1] One need not understand the musical notation to make sense of this example. Simply consider each note to be a unique symbol in a set of symbols having a fixed ordering. The example is framed in terms of music because my original work involved music composition.

## References

Hinton, G. E. (1988). Representing part–whole hierarchies in connectionist networks. *Proceedings of the Eighth Annual Conference of the Cognitive Science Society.*

Jordan, M. I. (1987). Attractor dynamics and parallelism in a connectionist sequential machine. In *Proceedings of the Eighth Annual Conference of the Cognitive Science Society* (pp. 531–546). Hillsdale, NJ: Erlbaum.

McClelland, J. L. (1979). On the time relations of mental processes: An examination of systems of processes in cascade. *Psychological Review, 86,* 287–330.

Miyata, Y., & Burr, D. (1990). Hierarchical recurrent networks for learning musical structure. Unpublished Manuscript.

Mozer, M. C. (1989). A focused back-propagation algorithm for temporal pattern recognition. *Complex Systems, 3,* 349–381.

Mozer, M. C., & Soukup, T. (1991). CONCERT: A connectionist composer of erudite tunes. In R. P. Lippmann, J. Moody, & D. S. Touretzky (Eds.), *Advances in neural information processing systems 3* (pp. 789–796). San Mateo, CA: Morgan Kaufmann.

Pearlmutter, B. A. (1989). Learning state space trajectories in recurrent neural networks. *Neural Computation, 1,* 263–269.

Pineda, F. (1987). Generalization of back propagation to recurrent neural networks. *Physical Review Letters, 19,* 2229–2232.

Rohwer, R. (1990). The 'moving targets' training algorithm. In D. S. Touretzky (Ed.), *Advances in neural information processing systems 2* (pp. 558–565). San Mateo, CA: Morgan Kaufmann.

Rumelhart, D. E., Hinton, G. E., & Williams, R. J. (1986). Learning internal representations by error propagation. In D. E. Rumelhart & J. L. McClelland (Eds.), *Parallel distributed processing: Explorations in the microstructure of cognition. Volume I: Foundations* (pp. 318–362). Cambridge, MA: MIT Press/Bradford Books.

Schmidhuber, J. (1991). *Neural sequence chunkers* (Report FKI-148-91). Munich, Germany: Technische Universitaet Muenchen, Institut fuer Informatik.

Williams, R. J., & Zipser, D. (1989). A learning algorithm for continually running fully recurrent neural networks. *Neural Computation, 1,* 270–280.
